# The Rescorla-Wagner algorithm and Maximum Likelihood estimation of causal parameters.

**Alan Yuille**
Department of Statistics
University of California at Los Angeles
Los Angeles, CA 90095
yuille@stat.ucla.edu

## Abstract

This paper analyzes generalization of the classic Rescorla-Wagner (R-W) learning algorithm and studies their relationship to Maximum Likelihood estimation of causal parameters. We prove that the parameters of two popular causal models, $\Delta P$ and $PC$, can be learnt by the same generalized linear Rescorla-Wagner (GLRW) algorithm provided genericity conditions apply. We characterize the fixed points of these GLRW algorithms and calculate the fluctuations about them, assuming that the input is a set of i.i.d. samples from a fixed (unknown) distribution. We describe how to determine convergence conditions and calculate convergence rates for the GLRW algorithms under these conditions.

## 1  Introduction

There has recently been growing interest in models of causal learning formulated as probabilistic inference [1,2,3,4,5]. There has also been considerable interest in relating this work to the Rescorla-Wagner learning model [3,5,6] (also known as the delta rule). In addition, there are studies of the equilibria of the Rescorla-Wagner model [6].

This paper proves mathematical results about these related topics. In Section (2), we describe two influential models, $\Delta P$ and $PC$, for causal inference and how their parameters can be learnt by maximum likelihood estimation from training data. Section (3) introduces the generalized linear Rescorla-Wagner (GLRW) algorithm, characterize its fixed points and quantify its fluctuations. We demonstrate that a simple GLRW can estimate the M-L parameters for both the $\Delta P$ and $PC$ models provided certain genericity conditions are satisfied. But the experimental conditions studied by Cheng [2] require a non-linear generalization of Rescorla-Wagner (Yuille, in preparation). Section (4) gives a way to determine convergence conditions and calculate the convergence rates of GLRW algorithms. Finally Section (5) sketches how the results in this paper can be extended to allow for arbitrary number of causes.

## 2 Causal Learning and Probabilistic Inference

The task is to estimate the causal effect of variables. There is an observed event $E$ and two causes $C_1, C_2$. Observers are asked to determine the *causal power* of the two causes. The variables are binary-valued. $E = 1$ means the event occurs, $E = 0$ means it does not. Similarly for causes $C_1$ and $C_2$. Much of the work in this section can be generalized to cases where there are an arbitrary number of causes $C_1, C_2, ..., C_N$, see section (5). The training data $\{(E^\mu, C_1^\mu, C_2^\mu)\}$ is assumed to be samples from an unknown distribution $P_{emp}(E, C_1, C_2)$.

Two simple models, $\Delta P$ [1] and $PC$ [2,3], have been proposed to account for how people estimate causal power. There is also a more recent theory based on model selection [4].

The $\Delta P$ and $PC$ theories are equivalent to assuming probability distributions for how the training data is generated. Then the *power* of the causes is given by the maximum likelihood estimation of the distribution parameters $\omega_1, \omega_2$. The two theories correspond to probability distributions $P_{\Delta P}(E|C_1, C_2, \omega_1, \omega_2)$ and $P_{PC}(E|C_1, C_2, \omega_1, \omega_2)$ given by:

$$P_{\Delta P}(E = 1|C_1, C_2, \omega_1, \omega_2) = \omega_1 C_1 + \omega_2 C_2. \; \Delta P \text{ model.} \tag{1}$$

$$P_{PC}(E = 1|C_1, C_2, \omega_1, \omega_2) = \omega_1 C_1 + \omega_2 C_2 - \omega_1 \omega_2 C_1 C_2. \; PC \text{ model.} \tag{2}$$

The later is a noisy-or model. The event $E = 1$ can be caused by $C_1 = 1$ with probability $\omega_1$, by $C_2 = 1$ with probability $\omega_2$, or caused by both. The model can be derived by setting $P_{PC}(E = 0|C_1, C_2, \omega_1, \omega_2) = (1 - \omega_1 C_1)(1 - \omega_2 C_2)$.

We assume that there is also a distribution on the causes $P(C_1, C_2|\vec{\gamma})$ which the observers also learn from the training data. This is equivalent to maximizing (with respect to $\omega_1, \omega_2, \vec{\gamma}$)):

$$P(\{(E^\mu, \vec{C}^\mu)\} : \vec{\omega}, \vec{\gamma}) = \prod_\mu P(E^\mu, \vec{C}^\mu : \vec{\omega}, \vec{\gamma}) = \prod_\mu P(E^\mu|\vec{C}^\mu : \vec{\omega})P(\vec{C}^\mu : \vec{\gamma}). \tag{3}$$

By taking logarithms, we see that estimating $\omega_1, \omega_2$ and $\vec{\gamma}$ are independent. So we will concentrate on estimating the $\omega_1, \omega_2$.

If the training data $\{E^\mu, \vec{C}^\mu\}$ is consistent with the model – i.e. there exist parameters $\omega_1, \omega_2$ such $P_{emp}(E|C_1, C_2) = P(E|C_1, C_2, \omega_1, \omega_2^)$ – then we can calculate the solution directly.

For the $\Delta P$ model, we have:

$$\omega_1 = P_{emp}(E = 1|C_1 = 1, C_2 = 0) = P_{emp}(E = 1|C_1 = 1),$$
$$\omega_2 = P_{emp}(E = 1|C_1 = 0, C_2 = 1) = P_{emp}(E = 1|C_2 = 1). \tag{4}$$

For the $P_{PC}$ model, we obtain Cheng's measures of causality [2,3].

$$\omega_1 = \frac{P_{emp}(E = 1|C_1 = 1, C_2) - P_{emp}(E = 1|C_1 = 0, C_2)}{1 - P_{emp}(E = 1|C_1 = 0, C_2)\}}$$

$$\omega_2 = \frac{P_{emp}(E = 1|C_1, C_2 = 1) - P_{emp}(E = 1|C_1, C_2 = 0)}{1 - P_{emp}(E = 1|C_1, C_2 = 0)\}}. \tag{5}$$

## 3 Generalized Linear Rescorla-Wagner

The Rescorla-Wagner model [7] is an alternative way to account for human learning. This iterative algorithm specifies an update rule for weights. These weights could measure the strength of a cause, such as the parameters of the Maximum Likelihood estimation. Following recent work [3,6], we seek to find relationships between generalized linear Rescorla-Wagner (GLRW) and ML estimation.

### 3.1 GLRW and two special cases

The Rescorla-Wagner algorithm updates weights $\{\vec{V}\}$ using training data $\{E^\mu, \vec{C}^\mu\}$. It is of form:

$$\vec{V}^{t+1} = \vec{V}^t + \Delta\vec{V}^t. \tag{6}$$

In this paper, we are particularly concerned with two special cases for choice of the update $\Delta V$.

$$\Delta V_1 = \alpha_1 C_1(E - C_1 V_1 - C_2 V_2), \;\; \Delta V_2 = \alpha_2 C_2(E - C_1 V_1 - C_2 V_2), \text{ basic} \tag{7}$$

$$\Delta V_1 = \alpha_1 C_1(1 - C_2)(E - V_1), \;\; \Delta V_2 = \alpha_2 C_2(1 - C_1)(E - V_2), \text{ variant.} \tag{8}$$

The first (7) is the *basic RW algorithm*. The second (8) is a *variant of RW* with a natural interpretation – a weight $V_1$ is updated only if one cause is present, $C_1 = 1$, and the other cause is absent, $C_2 = 0$.

The most general GLRW is of form:

$$\Delta V_i^t = \sum_{j=1}^{N} V_j^t f_{ij}(E^t, \vec{C}^t) + g_i(E^t, \vec{C}^t), \; \forall i, \tag{9}$$

where $\{f_{ij}(.,.) : i, j = 1, ..., N\}$ and $\{g_i(.) : i = 1, ..., N\}$ are functions of the data samples $E^\mu, \vec{C}^\mu$.

### 3.2 GLRW and Stochastic Samples

Our analysis assumes that the data samples $\{E^\mu, \vec{C}^\mu)\}$ are independent identical (i.i.d.) samples from an unknown distribution $P_{emp}(E|\vec{C})P(\vec{C})$.

In this case, the GLRW becomes stochastic. It defines a distribution on weights which is updated as follows:

$$P(\vec{V}^{t+1}|\vec{V}^t) = \int dE^t \, d\vec{C}^t \prod_{i=1}^{N} \delta(V_i^{t+1} - V_i^t - \Delta V_i^t) P(E^t, \vec{C}^t). \tag{10}$$

This defines a Markov Chain. If certain conditions are satisfied (see section (4), the chain will converge to a fixed distribution $P^*(V)$. This distribution can be characterized by its expected mean $< V >^* = \sum_V V P^*(V)$ and its expected covariance $\Sigma^* = \sum_V (V - < V >^*)(V - < V >^*)^T P^*(V)$. In other words, even after convergence the weights will fluctuate about the expected mean $< V >^*$ and the magnitude of the fluctuations will be given by the expected covariance.

### 3.3 What Does GLRW Converge to?

We now compute the means and covariance of the fixed point distribution $P^*(\vec{V})$. We first do this for the GLRW, equation (9), and then we restrict ourselves to the two special cases, equations (7,8).

**Theorem 1.** *The means $\vec{V}^*$ and the covariance $\Sigma^*$ of the fixed point distribution $P^*(\vec{V})$, using the GLRW equation (9) and any empirical distribution $P_{emp}(E, \vec{C})$ are given by the solutions to the linear equations,*

$$\sum_{j=1}^{N} V_j^* \sum_{E,\vec{C}} f_{ij}(E, \vec{C}) P_{emp}(E, \vec{C}) + \sum_{E,\vec{C}} g_i(E, \vec{C}) P_{emp}(E, \vec{C}) = 0, \; \forall i, \tag{11}$$

*and $\forall i, j$:*

$$\Sigma_{ik}^* = \sum_{jl} \Sigma_{jl}^* \sum_{E,\vec{C}} A_{ij}(E,\vec{C}) A_{kl}(E,\vec{C}) P_{emp}(E,\vec{C})$$
$$+ \sum_{E,\vec{C}} B_i(E,\vec{C}) B_k(E,\vec{C}) P_{emp}(E,\vec{C}), \qquad (12)$$

*where $A_{ij}(E,\vec{C}) = \delta_{ij} + f_{ij}(E,\vec{C})$ and $B_i(E,\vec{C}) = \sum_j V_j^* f_{ij}(E,\vec{C}) + g_i(E,\vec{C})$ (here $\delta_{ij}$ is the Kronecker delta function defined by $\delta_{ij} = 1$, if $i = j$ and $= 0$ otherwise). The means have a unique solution provided $\sum_{E,\vec{C}} P_{emp}(E,\vec{C}) f_{ij}(E,\vec{C})$ is an invertible matrix.*

Proof. *We derive the formula for the means $\vec{V}^*$ by taking the expectation of the update rule, see equations (9), with respect to $P^*(\vec{V})$ and $P_{emp}(E,\vec{C})$. To calculate the covariances, we express the update rule as:*

$$V_i^{t+1} - V_i^* = \sum_j (V_j^t - V_j^*) A_{ij}(E,\vec{C}) + B_i(E,\vec{C}), \ \forall i \qquad (13)$$

*with $A_{ij}(E,\vec{C})$ and $B_i(E,\vec{C})$ defined as above. Then we multiply both sides of equation (13) by their transpose (e.g. the left hand side by $(V_k^{t+1} - V_k^*)$) and taking the expectation with respect to $P^*(\vec{V})$ and $P_{emp}(E,\vec{C})$ (making use of the result that the expected value of $V_j^t - V_j^*$ is zero as $t \to \infty$.*

We can apply these results to study the behaviour of the two special cases, equations (7,8), when the data is generated by either the $\Delta P$ or $PC$ model.

First consider the basic RW algorithm (7) when the data is generated by the $P_{\Delta P}$ model. We can use Theorem 1 to rederive the result that $< \vec{V} >^* = \vec{\omega}$ [3,6], and so basic RW performs ML estimation for the $P_{\Delta P}$ model. It also follows directly. that if the data is generated by the $P_{PC}$ model, then $< \vec{V} >^* \neq \vec{\omega}$ (although they are related by a nonlinear equation).

Now consider the variant RW, equation (8).

**Theorem 2**. *The expected means of the fixed points of the variant RW equation (8) when the data is generated by probability model $P_{PC}(E|\vec{C},\vec{\omega})$ or $P_{\Delta P}(E|\vec{C};\vec{\omega})$ are given by:*

$$V_1^* = \omega_1, V_2^* = \omega_2, \qquad (14)$$

*provided $P_{emp}(\vec{C})$ satisfies genericity conditions so that $< C_1(1-C_2) > < C_2(1-C_1) > \neq 0$.*

*The expected covariances are given by:*

$$\Sigma_{11} = \omega_1(1-\omega_1)\frac{\alpha_1}{2-\alpha_1}, \Sigma_{22} = \omega_2(1-\omega_2)\frac{\alpha_2}{2-\alpha_2}, \Sigma_{12} = \Sigma_{21} = 0. \qquad (15)$$

. Proof. *This is a direct calculation of quantities specified in Theorem 1. For example, we calculate the expected value of $\Delta V_1$ and $\Delta V_2$ first with respect to $P(E|\vec{C})$ and then with respect to $P^*(V)$. This gives:*

$$< \Delta V_1 >_{P(E|\vec{C})P^*(V)} = \alpha_1 C_1(1-C_2)(\omega_1 - V_1^*),$$
$$< \Delta V_2 >_{P(E|\vec{C})P^*(V)} = \alpha_2 C_2(1-C_1)(\omega_2 - V_2^*), \qquad (16)$$

*where we have used $\sum_V P^*(V)V = V^*$, $\sum_E E P_{PC}(E|\vec{C}) = \omega_1 C_1 + \omega_2 C_2 - \omega_1 \omega_2 C_1 C_2$, and logical relations to simply the terms (e.g. $C_1^2 = C_1$, $C_1(1-C_1) = 0$).*

*Taking the expectation of $< \Delta V_1 >_{P(E|\vec{C})P^*(V)}$ with respect to $P(C)$ gives,*

$$\alpha_1 \omega_1 < C_1(1 - C_2) >_{P(C)} -\alpha_1 V_1^* < C_1(1 - C_2) >= 0,$$
$$\alpha_2 \omega_2 < C_2(1 - C_1) >_{P(C)} -\alpha_2 V_2^* < C_2(1 - C_1) >= 0, \tag{17}$$

*and the result follows directly, except for non-generic cases where $< C_1(1 - C_2) >= 0$ or $< C_2(1 - C_1) >= 0$. These degenerate cases are analyzed separately.*

It is perhaps surprising that the same GLRW algorithm can perform ML estimation when the data is generated by either model $P_{\Delta P}$ or $P_{PC}$ (and this can be generalized, see section (5)). Moreover, the expected covariance is the same for both models. Observe that the covariance decreases if we make the update coefficients $\alpha_1, \alpha_2$ of the algorithm small. The convergence rates are given in the next section.

The non-generic cases include the situation studied in [2] where $C_1$ is a background cause that it assumed to be always present, so $< C_1 >= 1$. In this case $V_1^* = \omega_1$, but $V_2^*$ is unspecified. It can be shown (Yuille, in preparation) that a nonlinear generalization of RW can perform ML on this problem (but it is eay to check that no GLRW can). But an even more ambiguous case occurs when $\omega_1 = 1$ (i.e. cause $C_1$ always causes event $E$), then there is no way to estimate $\omega_2$ and Cheng's measure of causality, equation (5), becomes undefined.

## 4   Convergence of Rescorla-Wagner

We now analyze the convergence of the GLRW algorithm. We obtain conditions for the algorithm to converge and give the convergence rates. For simplicity, the results will be illustrated only on the simple models.

Our results are based on the following theorem for the convergence of the state vector of a stochastic iterative equation. The theorem gives necessary and sufficient conditions for convergence, shows what the expected state vector converges to, and gives the rate of convergence.

**Theorem 3**. *Let $\vec{z}_{t+1} = \mathbf{A}_t \vec{z}_t$ be an iterative update equation, where $\vec{z}$ is a state vector and the update matrices $\mathbf{A}_t$ are i.i.d. samples from $P(\mathbf{A})$. The convergence properties as $t \to \infty$ depends on $< \mathbf{A} >= \sum_{\mathbf{A}} \mathbf{A} P(\mathbf{A})$. If $< \mathbf{A} >$ has a unit eigenvalue with eigenvector $\vec{z}^*$ and the next largest eigenvalue has modulus $\lambda < 1$, then $\lim_{t \to \infty} < \vec{z}_t > \propto \vec{z}^*$ and the rate of convergence is $e^{t \log \lambda}$. If the moduli of the eigenvalues of $< \mathbf{A} >$ are all less than 1, then $\lim_{t \to \infty} < \vec{z}_t >= 0$. If $< \mathbf{A} >$ has an eigenvalue with modulus greater than 1, then $< \vec{z}_t >$ diverges as $t \to \infty$.*

*Proof. This is a standard result. To obtain it, write $\vec{z}_{t+1} = \mathbf{A}_t \mathbf{A}_{t-1}....\mathbf{A}_1 \vec{z}_1$, where $\vec{z}_1$ is the initial condition. Now take the expectation of $\vec{z}_{t+1}$ with respect to the samples $\{(a_t, b_t)\}$. By the i.i.d. assumption, this gives $< \vec{z}_{t+1} >= < \mathbf{A} >^t \vec{z}_1$. The result follows by linear algebra. Let the eigenvectors and eigenvalues of $< \mathbf{A} >$ be $\{(\lambda_i, \vec{e}_i)\}$. Express the initial conditions as $\vec{z}_1 = \sum \gamma_i \vec{e}_i$ where the $\{\gamma_i\}$ are coefficients. Then $< \vec{z}_t >= \sum_i \gamma_i \lambda^t \vec{e}_i$, and the result follows.*

We use Theorem 3 to obtain convergence results for the GLRW algorithm. To ensure convergence, we need both the expected covariance and the expected means to converge. Then Markov's lemma can be used to bound the fluctuations. (If we just require the expected means to converge, then the fluctuations of the weights may be infinitely large). This can be done by a suitable choice of the state vector $\vec{z}$.

For simplicity of algebra, we demonstrate this for a GLRW algorithm with a single weight. The update rule is $V_{t+1} = a_t V_t + b_t$ where $a_t, b_t$ are random samples. We define the state vector to be $\vec{z} = (V_t^2, V_t, 1)$.

**Theorem 4.** *Consider the stochastic update rule $V_{t+1} = a_t V_t + b_t$ where $a_t$ and $b_t$ are samples from distributions $P_a(a)$ and $P_b(b)$. Define $\alpha_1 = \sum_a a^2 P(a)$, $\alpha_2 = \sum_a a P(a)$, $\beta_1 = \sum_b b^2 P(b)$, $\beta_2 = \sum_b b P(b)$, and $\gamma = 2 \sum_{a,b} ab P(a,b)$. The algorithm converges if, and only if, $\alpha_1 < 1, \alpha_2 < 1$. If so, then $\lim_{t \to \infty} < V_t > = < V > = \frac{\beta_2}{1-\alpha_2}$, $\lim_{t \to \infty} < (V_t - < V >)^2 > = \frac{\beta_1(1-\alpha_2)+\gamma\beta_2}{(1-\alpha_1)(1-\alpha_2)} - \frac{\beta_2^2}{(1-\alpha_2)^2}$. The convergence rate is $\{\max\{\alpha_1, |\alpha_2|\}\}^t$.*

Proof. *Define $\vec{z}_t = (V_t^2, V_t, 1)$ and express the update rule in matrix form:*

$$\begin{pmatrix} V_{t+1}^2 \\ V_{t+1} \\ 1 \end{pmatrix} = \begin{pmatrix} a_t^2 & 2a_t b_t & b_t^2 \\ 0 & a_t & b_t \\ 0 & 0 & 1 \end{pmatrix} \begin{pmatrix} V_t^2 \\ V_t \\ 1 \end{pmatrix}$$

*This is of the form analyzed in Theorem 3 provided we set:*

$$\mathbf{A} = \begin{pmatrix} a_t^2 & 2a_t b_t & b_t^2 \\ 0 & a_t & b_t \\ 0 & 0 & 1 \end{pmatrix} \text{ and } < \mathbf{A} > = \begin{pmatrix} \alpha_1 & \gamma & \beta_1 \\ 0 & \alpha_2 & \beta_2 \\ 0 & 0 & 1 \end{pmatrix},$$

*where $\alpha_1 = \sum_a a^2 P(a)$, $\alpha_2 = \sum_a a P(a)$, $\beta_1 = \sum_b b^2 P(b)$, $\beta_2 = \sum_b b P(b)$, and $\gamma = 2 \sum_{a,b} ab P(a,b)$.*

*The eigenvalues $\{\lambda\}$ and eigenvectors $\{\vec{e}\}$ of $< \mathbf{A} >$ are:*

$$\lambda_1 = 1, \ \vec{e}_1 \propto \left(\frac{\beta_1(1-\alpha_2)+\gamma\beta_2}{(1-\alpha_1)(1-\alpha_2)}, \frac{\beta_2}{1-\alpha_2}, 1\right)$$

$$\lambda_2 = \alpha_1, \ \vec{e}_2 = (1,0,0), \quad \lambda_3 = \alpha_2, \ \vec{e}_3 \propto \left(\frac{\gamma}{\alpha_2 - \alpha_1}, 1, 0\right). \tag{18}$$

*The result follows from Theorem 3.*

Observe that if $|\alpha_2| < 1$ but $\alpha_1 > 1$, then $< V_t >$ will converge but the expected variance does not. The fluctuations in the GLRW algorithm will be infinitely large.

We can extend Theorem 4 to the variant of RW equation (8). Let $P = P_{emp}$, then

$$\beta_{12} = \sum_{E,\vec{C}} P(E|\vec{C})P(\vec{C})C_1(1-C_2), \quad \beta_{21} = \sum_{E,\vec{C}} P(E|\vec{C})P(\vec{C})C_2(1-C_1),$$

$$\gamma_{12} = \sum_{E,\vec{C}} P(E|\vec{C})P(\vec{C})EC_1(1-C_2), \quad \gamma_{21} = \sum_{E,\vec{C}} P(E|\vec{C})P(\vec{C})EC_2(1-C_1). \tag{19}$$

If the data is generated by $P_{\Delta P}$ or $P_{PC}$, then $\beta_{12}, \beta_{21}, \gamma_{12}, \gamma_{21}$ take the same values:

$$\beta_{12} = < C_1(1-C_2) >, \ \beta_{21} = < (1-C_1)C_2 >,$$

$$\gamma_{12} = \omega_1 < C_1(1-C_2) >, \ \gamma_{21} = \omega_2 < (1-C_1)C_2 > . \tag{20}$$

**Theorem 5.** *The algorithm specified by equation (8) converges provided $\lambda^* = \max\{|\lambda_2|, |\lambda_3|, |\lambda_4|, |\lambda_5|\} < 1$, where $\lambda_2 = 1 - (2\alpha_1 - \alpha_1^2)\beta_{12}, \ \lambda_3 = 1 - (2\alpha_2 - \alpha_2^2)\beta_{21}$, $\lambda_4 = 1 - \alpha_1 \beta_{12} \ \lambda_5 = 1 - \alpha_2 \beta_{21}$. The convergence rate is $e^{t \log \lambda^*}$. The expected means and covariances can be calculated from the first eigenvector.*

Proof. *We define the state vector $\vec{z} = (V_1^2, V_2^2, V_1, V_2, 1)$ and derive the update matrix $\mathbf{A}$ from equation (8). The eigenvectors and eigenvalues can be calculated (calculations omitted due to space constraints). The eigenvalues are $1, \lambda_1, \lambda_2, \lambda_3, \lambda_4$. The convergence conditions and rates follow from Theorem 3. The expected means and covariances can be calculated from the first eigenvector, which is:*

$$\vec{e}_1 = \left(\frac{2(\alpha_1 - \alpha_1^2)\gamma_{12}^2}{(2\alpha_1 - \alpha_1^2)\beta_{12}^2} + \frac{\alpha_1^2 \gamma_{12}}{(2\alpha_1 - \alpha_1^2)\beta_{12}}, \frac{2(\alpha_2 - \alpha_2^2)\gamma_{21}^2}{(2\alpha_2 - \alpha_2^2)\beta_{21}^2} + \frac{\alpha_2^2 \gamma_{21}}{(2\alpha_2 - \alpha_2^2)\beta_{21}}, \frac{\gamma_{12}}{\beta_{12}}, \frac{\gamma_{21}}{\beta_{21}}, 1\right), \tag{21}$$

*and they agree with the calculations given in Theorem 2.*

## 5 Generalization

The results of the previous sections can be generalized to cases where there are more than two causes. For example, we can use the generalization of the $PC$ model to include multiple *generative causes* $\vec{C}$ and *preventative causes* $\vec{L}$, [5] extending [2].

The probability distribution for this generalized $PC$ model is:

$$P_{PC}(E = 1 | \vec{C}, \vec{L}; \vec{\omega}, \vec{\Omega}) = \{1 - \prod_{i=0}^{n}(1 - \omega_i C_i)\} \prod_{j=1}^{m}(1 - \Omega_j L_j), \qquad (22)$$

where there are $n+1$ *generative causes* $\{C_i\}$ and $m$ *preventative causes* $\{L_j\}$ specified in terms of parameters $\{\omega_i\}$ and $\{\Omega_j\}$ (constrained to lie between 0 and 1).

We assume that there is a single background cause $C_0$ which is always on (i.e. $C_0 = 1$) and whose strength $\omega_0$ is known (for relaxing this constraint, see Yuille in preparation).

Then it can be shown that the following GLRW algorithm will converge to the ML estimates of the remaining parameters $\{\omega_1 : 1 = 1, ..., n\}$ and $\{\Omega_j : j = 1, ..., m\}$ of the generalized $PC$ model:

$$\begin{aligned} \Delta V_k^t &= C_k \{\prod_{i=1}^{m}(1 - L_i) \prod_{j=1:j\neq k}^{n}(1 - C_j)\}(E - \omega_0 - (1 - \omega_0)V_k^t), \\ \Delta U_l^t &= L_l \{\prod_{k=1:k\neq l}^{m}(1 - L_k) \prod_{j=1}^{n}(1 - C_j)\}(E - \omega_0 - \omega_0 U_l^t), \qquad (23) \end{aligned}$$

where $\{V_k : k = 1, ..., n\}$ and $\{U_l : l = 1, ..., m\}$ are weights.

The proof is straightforward algebra and is based on the following identity for binary variables: $\prod_j(1 - \Omega_j L_j) \prod_j(1 - L_j) = \prod_j(1 - L_j)$.

The GLRW algorithm (23) will also perform ML estimation for data generated by other probability distributions which share the same linear terms as the generalized $PC$ model (i.e. the terms linear in the $\{\omega_i\}$ and $\{\Omega_j\}$.) The convergence conditions and the convergence rates can be calculated using the techniques in section (4).

These results all assume genericity conditions so that none of the generative or preventative causes is either always on or always off (i.e. ruling out case like [2]).

## 6 Conclusion

This paper introduced and studied generalized linear Rescorla-Wagner (GLRW) algorithms. We showed that two influential theories, $\Delta P$ and $PC$, for estimating causal effects can be implemented by the same GLRW, see (8). We obtained convergence results for GLRW including classifying the fixed points, calculating the asymptotic fluctuations, and the convergence rates. Our results assume that the GLRW are i.i.d. samples from an unknown empirical distribution $P_{emp}(E, \vec{C})$. Observe that the fluctuations of GLRW can be removed by introducing damping coefficients which decrease over time. Stochastic approximation theory [8] can then be used to give conditions for convergence.

More recent work (Yuille in preparation) clarifies the class of maximum likelihood inference problems that can be "solved" by GLRW and by non-linear GLRW. In particular, we show that a non-linear RW can perform ML estimation for the non-generic case studied by Cheng. We also investigate similarities to Kalman filter models [9].

## Acknowledgements

I thank Patricia Cheng, Peter Dayan and Yingnian Wu for helpfull discussions. Anonymous referees gave useful feedback that has motivated a follow-up paper. This work was partially supported by an NSF SLC catalyst grant "Perceptual Learning and Brain Plasticity" NSF SBE-0350356.

## References

[1]. B. A. Spellman. "Conditioning Causality". In D.R. Shanks, K.J. Holyoak, and D.L. Medin, (eds). **Causal Learning: The Psychology of Learning and Motivation, Vol. 34**. San Diego, California. Academic Press. pp 167-206. 1996.

[2]. P. Cheng. "From Covariance to Causation: A Causal Power Theory". *Psychological Review*, **104**, pp 367-405. 1997.

[3]. M. Buehner and P. Cheng. "Causal Induction: The power PC theory versus the Rescorla-Wagner theory". In *Proceedings of the 19th Annual Conference of the Cognitive Science Society"*. 1997.

[4]. J.B. Tenenbaum and T.L. Griffiths. "Structure Learning in Human Causal Induction". Advances in Neural Information Processing Systems 12. MIT Press. 2001.

[5]. D. Danks, T.L. Griffiths, J.B. Tenenbaum. "Dynamical Causal Learning". *Advances in Neural Information Processing Systems 14*. 2003.

[6]. D. Danks. "Equilibria of the Rescorla-Wagner Model". *Journal of Mathematical Psychology*. Vol. 47, pp 109-121. 2003.

[7]. R.A. Rescorla and A.R. Wagner. "A Theory of Pavlovian Conditioning: Variations in the Effectiveness of Reinforcement and Nonreinforcement". In A.H. Black andW.F. Prokasy, eds. **Classical Conditioning II: Current Research and Theory.** New York. Appleton-Century-Crofts, pp 64-99. 1972.

[8]. H.J. Kushner and D.S. Clark. **Stochastic Approximation for Constrained and Unconstrained Systems.** New York. Springer-Verlag. 1978.

[9]. P. Dayan and S. Kakade. "Explaining away in weight space". In *Advances in Neural Information Processing Systems 13*. 2001.
